# The Kernel Gibbs Sampler

**Thore Graepel**
Statistics Research Group
Computer Science Department
Technical University of Berlin
Berlin, Germany
*guru@cs.tu-berlin.de*

**Ralf Herbrich**
Statistics Research Group
Computer Science Department
Technical University of Berlin
Berlin, Germany
*ralfh@cs.tu-berlin.de*

## Abstract

We present an algorithm that samples the hypothesis space of kernel classifiers. Given a uniform prior over normalised weight vectors and a likelihood based on a model of label noise leads to a piecewise constant posterior that can be sampled by the kernel Gibbs sampler (KGS). The KGS is a Markov Chain Monte Carlo method that chooses a random direction in parameter space and samples from the resulting piecewise constant density along the line chosen. The KGS can be used as an analytical tool for the exploration of Bayesian transduction, Bayes point machines, active learning, and evidence-based model selection on small data sets that are contaminated with label noise. For a simple toy example we demonstrate experimentally how a Bayes point machine based on the KGS outperforms an SVM that is incapable of taking into account label noise.

## 1 Introduction

Two great ideas have dominated recent developments in machine learning: the application of *kernel methods* and the popularisation of *Bayesian inference.* Focusing on the task of classification, various connections between the two areas exist: kernels have long been a part of Bayesian inference in the disguise of covariance functions that characterise priors over functions [9]. Also, attempts have been made to re-derive the support vector machine (SVM) [1] — possibly the most prominent representative of kernel methods — as a maximum a-posteriori estimator (MAP) in a Bayesian framework [8]. While this work suggests good strategies for evidence-based model selection the MAP estimator is not truly Bayesian in spirit because it is not based on the concept of model averaging which is crucial to Bayesian reasoning. As a consequence, the MAP estimator is generally not as robust as a real Bayesian estimator. While this drawback is inconsequential in a noise-free setting or in a situation dominated by *feature noise,* it may have severe consequences when the data is contaminated by *label noise* that may lead to a multi-modal posterior distribution. In order to make use of the full Bayesian posterior distribution it is necessary to generate samples from this distribution. This contribution is concerned with the generation of samples from the Bayesian posterior over the hypothesis space of lin-

ear classifiers in arbitrary kernel spaces in the case of label noise. In contrast to [8] we consider normalised weight vectors, $\|\mathbf{w}\|_{\mathcal{K}} = 1$, because the classification given by a linear classifier only depends on the *spatial direction* of the weight vector $\mathbf{w}$ and not on its length. This point of view leads to a hypothesis space isomorphic to the surface of an $n$-dimensional sphere which — in the absence of prior information — is naturally equipped with a uniform prior over directions. Incorporating the label noise model into the likelihood then leads to a piecewise constant posterior on the surface of the sphere. The *kernel Gibbs sampler* (KGS) is designed to sample from this type of posterior by iteratively choosing a random direction and sampling on the resulting piecewise constant one-dimensional density in the fashion of a hit-and-run algorithm [7].

The resulting samples can be used in various ways: i) In *Bayesian transduction* [3] the decision about the labels of new test points can be inferred by a majority decision of the sampled classifiers. ii) The posterior mean — the *Bayes point machine* (BPM) solution [4] — can be calculated as an approximation to transduction. iii) The binary entropy of candidate training points can be calculated to determine their information content for *active learning* [2]. iv) The *model evidence* [5] can be evaluated for the purpose of model selection. We would like to point out, however, that the KGS is limited in practice to a sample size of $m \approx 100$ and should thus be thought of as an analytical tool to advance our understanding of the interaction of kernel methods and Bayesian reasoning.

The paper is structured as follows: in Section 2 we introduce the learning scenario and explain our Bayesian approach to linear classifiers in kernel spaces. The kernel Gibbs sampler is explained in detail in Section 3. Different applications of the KGS are discussed in Section 4 followed by an experimental demonstration of the BPM solution based on using the KGS under label noise conditions. We denote $n$–tuples by italic bold letters (e.g. $\boldsymbol{x}$), vectors by roman bold letters (e.g. $\mathbf{x}$), random variables by sans serif font (e.g. $\mathsf{X}$), and vector spaces by calligraphic capitalised letters (e.g. $\mathcal{X}$). The symbols $\mathbf{P}, \mathbf{E}$ and $\mathbf{I}$ denote a probability measure, the expectation of a random variable and the indicator function, respectively.

## 2 Bayesian Learning in Kernel spaces

We consider learning given a sequence $\boldsymbol{x} = (x_1, \ldots, x_m) \in \mathcal{X}^m$ and $\boldsymbol{y} = (y_1, \ldots y_m) \in \{-1, +1\}^m$ drawn iid from a fixed distribution $\mathbf{P}_{\mathsf{XY}} = \mathbf{P}_{\mathsf{Z}}$ over the space $\mathcal{X} \times \{-1, +1\} = \mathcal{Z}$ of input-output pairs. The hypotheses are linear classifiers $x \mapsto \langle \mathbf{w}, \phi(x) \rangle_{\mathcal{K}} =: \langle \mathbf{w}, \mathbf{x} \rangle_{\mathcal{K}}$ in some fixed *feature space* $\mathcal{K} \subseteq \ell_2^n$ where we assume that a mapping $\phi : \mathcal{X} \to \mathcal{K}$ is chosen a priori[1]. Since all we need for learning is the real-valued output $\langle \mathbf{w}, \mathbf{x}_i \rangle_{\mathcal{K}}$ of the classifier $\mathbf{w}$ at the $m$ training points in $x_1, \ldots, x_m$ we can assume that $\mathbf{w}$ can be expressed as (see [9])

$$\mathbf{w} = \sum_{i=1}^{m} \alpha_i \mathbf{x}_i \,. \tag{1}$$

Thus, it suffices to learn the $m$ expansion coefficients $\boldsymbol{\alpha} \in \mathbb{R}^m$ rather than the $n$ components of $\mathbf{w} \in \mathcal{K}$. This is particularly useful if the dimensionality $\dim(\mathcal{K}) = n$ of the feature space $\mathcal{K}$ is much greater (or possibly infinite) than the number $m$ of training points. From (1) we see that all that is needed is the inner product function $k(x, x') = \langle \phi(x), \phi(x') \rangle_{\mathcal{K}}$ also known as the *kernel* (see [9] for a detailed introduction to the theory of kernels).

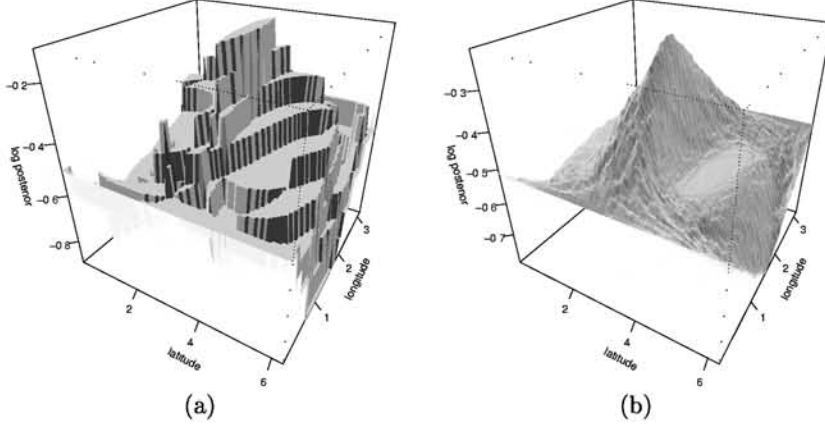

<div align="center">(a)　　　　　　　　　　　(b)</div>

Figure 1: Illustration of the (log) posterior distribution on the surface of a 3–dimensional sphere $\left\{ \mathbf{w} \in \mathbb{R}^3 \mid \|\mathbf{w}\|_{\mathcal{K}} = 1 \right\}$ resulting from a label noise model with a label flip rate of $q = 0.20$ **(a)** $m = 10$, **(b)** $m = 1\,000$. The log posterior is plotted over the longitude and latitude, and for small sample size it is multi-modal due to the label noise. The classifier $\mathbf{w}^*$ labelling the data (before label noise) was at $\left( \frac{\pi}{2}, \pi \right)$.

In a Bayesian spirit we consider a prior $\mathbf{P_W}$ over possible weight vectors $\mathbf{w} \in \mathcal{W}$ of unit length, i.e. $\mathcal{W} = \left\{ \mathbf{v} \in \mathcal{K} \mid \|\mathbf{v}\|_{\mathcal{K}} = 1 \right\}$. Given an iid training set $\boldsymbol{z} = (\boldsymbol{x}, \boldsymbol{y})$ and a likelihood model $\mathbf{P}_{\mathsf{Y}|\mathsf{X}=x,\mathbf{W}=\mathbf{w}}$ we obtain the posterior $\mathbf{P}_{\mathbf{W}|\mathsf{Z}^m=\boldsymbol{z}}$ using Bayes' formula

$$\mathbf{P}_{\mathbf{W}|\mathsf{Z}^m=\boldsymbol{z}}\left(\mathbf{w}\right) = \frac{\mathbf{P}_{\mathsf{Y}^m|\mathsf{X}^m=\boldsymbol{x},\mathbf{W}=\mathbf{w}}\left(\boldsymbol{y}\right)\mathbf{P_W}\left(\mathbf{w}\right)}{\mathbf{E_W}\left[\mathbf{P}_{\mathsf{Y}^m|\mathsf{X}^m=\boldsymbol{x},\mathbf{W}=\mathbf{w}}\left(\boldsymbol{y}\right)\right]}, \tag{2}$$

By the iid assumption and the independence of the denominator from $\mathbf{w}$ we obtain

$$\mathbf{P}_{\mathbf{W}|\mathsf{Z}^m=\boldsymbol{z}}\left(\mathbf{w}\right) \propto \underbrace{\prod_{i=1}^{m} \mathbf{P}_{\mathsf{Y}|\mathsf{X}=x_i,\mathbf{W}=\mathbf{w}}\left(y_i\right)}_{\mathcal{L}[\mathbf{w},\boldsymbol{z}]} \cdot \mathbf{P_W}\left(\mathbf{w}\right).$$

In the absence of specific prior knowledge symmetry suggests to take $\mathbf{P_W}$ uniform on $\mathcal{W}$. Furthermore, we choose the likelihood model

$$\mathbf{P}_{\mathsf{Y}|\mathsf{X}=x,\mathbf{W}=\mathbf{w}}\left(y\right) = \begin{cases} q & \text{if } y\left\langle \mathbf{w},\mathbf{x} \right\rangle_{\mathcal{K}} \leq 0 \\ 1-q & \text{otherwise} \end{cases},$$

where $q$ specifies the assumed *level of label noise*. Please note the difference to the commonly assumed model of *feature noise* which essentially assumes noise in the (mapped) input vectors $\mathbf{x}$ instead of the labels $y$ and constitutes the basis of the soft-margin SVM [1]. Thus the likelihood $\mathcal{L}\left[\mathbf{w},\boldsymbol{z}\right]$ of the weight vector $\mathbf{w}$ is given by

$$\mathcal{L}\left[\mathbf{w},\boldsymbol{z}\right] = q^{m \cdot R_{\text{emp}}[\mathbf{w},\boldsymbol{z}]} \left(1-q\right)^{m\left(1-R_{\text{emp}}[\mathbf{w},\boldsymbol{z}]\right)}, \tag{3}$$

where the training error $R_{\text{emp}}\left[\mathbf{w},\boldsymbol{z}\right]$ is defined as

$$R_{\text{emp}}\left[\mathbf{w},\boldsymbol{z}\right] = \frac{1}{m}\sum_{i=1}^{m}\mathbf{I}_{y_i\left\langle \mathbf{w},\mathbf{x}_i\right\rangle_{\mathcal{K}}\leq 0}.$$

Two data points $y_1\mathbf{x}_1$ and $y_2\mathbf{x}_2$ divide the space of normalised weight vectors $\mathbf{w}$ into four equivalence classes with different posterior density indicated by the gray shading. In each iteration, starting from $\mathbf{w}_{j-1}$ a random direction $\mathbf{v}$ with $\mathbf{v}\perp\mathbf{w}_{j-1}$ is generated. We sample from the piecewise constant density on the great circle determined by the plane defined by $\mathbf{w}_{j-1}$ and $\mathbf{v}$. In order to obtain $\zeta^*$, we calculate the $2m$ angles $\zeta_i$ where the training samples intersect with the circle and keep track of the number $m \cdot e_i$ of training errors for each region $i$.

Figure 2: Schematic view of the kernel Gibbs sampling procedure.

Clearly, the posterior $\mathbf{P}_{\mathbf{W}|Z^m=\mathbf{z}}$ is piecewise constant for all $\mathbf{w}$ with equal training error $R_{\mathrm{emp}}[\mathbf{w},\mathbf{z}]$ (see Figure 1).

## 3   The Kernel Gibbs Sampler

In order to sample from $\mathbf{P}_{\mathbf{W}|Z^m=\mathbf{z}}$ on $\mathcal{W}$ we suggest a Markov Chain sampling method. For a given value of $q$, the sampling scheme can be decomposed into the following steps (see Figure 2):

1. Choose an arbitrary starting point $\mathbf{w}_0 \in \mathcal{W}$ and set $j = 0$.

2. Choose a direction $\mathbf{v} \in \mathcal{W}$ in the tangent space $\left\{ \tilde{\mathbf{v}} \in \mathcal{W} \mid \langle \tilde{\mathbf{v}}, \mathbf{w}_j \rangle_{\mathcal{K}} = 0 \right\}$.

3. Calculate all $m$ hit points $\mathbf{b}_i \in \mathcal{W}$ from $\mathbf{w}$ in direction $\mathbf{v}$ with the hyperplane having normal $y_i\mathbf{x}_i$. Before normalisation, this is achieved by [4]

$$\mathbf{b}_i = \mathbf{w}_j - \frac{\langle \mathbf{w}_j, \mathbf{x}_i \rangle_{\mathcal{K}}}{\langle \mathbf{v}, \mathbf{x}_i \rangle_{\mathcal{K}}} \mathbf{v}.$$

4. Calculate the $2m$ angular distances $\zeta_i$ from the current position $\mathbf{w}_j$

$$\forall i \in \{1, \ldots, m\}: \quad \zeta_{2i-1} = -\mathrm{sign}\left( \langle \mathbf{v}, \mathbf{b}_i \rangle_{\mathcal{K}} \right) \arccos\left( \langle \mathbf{w}_j, \mathbf{b}_i \rangle_{\mathcal{K}} \right),$$
$$\forall i \in \{1, \ldots, m\}: \quad \zeta_{2i} = (\zeta_{2i-1} + \pi) \mod (2\pi).$$

5. Sort the $\zeta_i$ in ascending order, i.e. $\Pi : \{1, \ldots, 2m\} \to \{1, \ldots, 2m\}$ such that

$$\forall i \in \{2, \ldots, 2m\}: \quad \zeta_{\Pi(i-1)} \le \zeta_{\Pi(i)}.$$

6. Calculate the training errors $e_i$ of the $2m$ intervals $\left[ \zeta_{\Pi(i-1)}, \zeta_{\Pi(i)} \right]$ by evaluating

$$e_i = R_{\mathrm{emp}}\left[ \cos\left( \frac{\zeta_{\Pi(i+1)} - \zeta_{\Pi(i)}}{2} \right) \mathbf{w}_j - \sin\left( \frac{\zeta_{\Pi(i+1)} - \zeta_{\Pi(i)}}{2} \right) \mathbf{v}, \mathbf{z} \right].$$

Here, we used the shorthand notation $\zeta_{\Pi(2m+1)} = \zeta_{\Pi(1)}$.

7. Sample an angle $\zeta^*$ using the piecewise uniform distribution and (3).

8. Calculate a new sample $\mathbf{w}_{j+1}$ by $\mathbf{w}_{j+1} = \cos{(\zeta^*)}\, \mathbf{w}_j - \sin{(\zeta^*)}\, \mathbf{v}$.

9. Set $j \leftarrow j + 1$ and go back to step 2.

Since the algorithm is carried out in feature space $\mathcal{K}$ we can use

$$\mathbf{w} = \sum_{i=1}^{m} \alpha_i \mathbf{x}_i, \quad \mathbf{v} = \sum_{i=1}^{m} \nu_i \mathbf{x}_i, \quad \mathbf{b} = \sum_{i=1}^{m} \beta_i \mathbf{x}_i\,.$$

For the inner products and norms it follows that, e.g.

$$\langle \mathbf{w}, \mathbf{v} \rangle_{\mathcal{K}} = \boldsymbol{\alpha}' \mathbf{G} \boldsymbol{\nu}, \quad \|\mathbf{w}\|_{\mathcal{K}}^2 = \boldsymbol{\alpha}' \mathbf{G} \boldsymbol{\alpha}\,,$$

where the $m \times m$ matrix $\mathbf{G}$ is known as the *Gram matrix* and is given by

$$\mathbf{G}_{ij} = \langle \mathbf{x}_i, \mathbf{x}_j \rangle_{\mathcal{K}} = k\,(x_i, x_j)\ .$$

As a consequence the above algorithm can be implemented in arbitrary kernel spaces only making use of $k$.

## 4 Applications of the Kernel Gibbs Sampler

The kernel Gibbs sampler provides samples from the full posterior distribution over the hypothesis space of linear classifiers in kernel space for the case of label noise. These samples can be used for various tasks related to learning. In the following we will present a selection of these tasks.

**Bayesian Transduction**  Given a sample from the posterior distribution over hypotheses, a good strategy for prediction is to let the sampled classifiers vote on each new test data point. This mode of prediction is closest to the Bayesian spirit and has been shown for the zero-noise case to yield excellent generalisation performance [3]. Also the fraction of votes for the majority decision is an excellent indicator for the reliability of the final estimate: Rejection of those test points with the closest decision results in a great reduction of the generalisation error on the remaining test points $x$. Given the posterior $\mathbf{P}_{\mathbf{W}|Z^m=z}$ the transductive decision is

$$BT_{\mathbf{z}}\,(x) = \mathrm{sign}\,\left(\mathbf{E}_{\mathbf{W}|Z^m=z}\left[\mathrm{sign}\,(\langle \mathbf{W}, \mathbf{x} \rangle_{\mathcal{K}})\right]\right)\ . \tag{4}$$

In practice, this estimator is approximated by replacing the expectation $\mathbf{E}_{\mathbf{W}|Z^m=z}$ by a sum over the sampled weight vectors $\mathbf{w}_j$.

**Bayes Point Machines**  For classification, Bayesian Transduction requires the whole collection of sampled weight vectors $\mathbf{w}$ in memory. Since this may be impractical for large data sets we would like to derive a single classifier $\mathbf{w}$ from the Bayesian posterior. An excellent approximation of the transductive decision $BT_{\mathbf{z}}\,(x)$ by a single classifier is obtained by exchanging the expectation with the inner sign-function in (4). Then the classifier $h_{\mathrm{bp}}$ is given by

$$h_{\mathrm{bp}}\,(x) \quad = \quad \mathrm{sign}\,\left(\langle \mathbf{E}_{\mathbf{W}|Z^m=z}\,[\mathbf{W}], \mathbf{x} \rangle_{\mathcal{K}}\right) = \mathrm{sign}\,\left(\langle \mathbf{w}_{\mathrm{bp}}, \mathbf{x} \rangle_{\mathcal{K}}\right)\,, \tag{5}$$

where the classifier $\mathbf{w}_{\mathrm{bp}}$ is referred to as the Bayes point and has been shown to yield generalisation performance superior to the well-known support vector solution $\mathbf{w}_{\mathrm{SVM}}$, which — in turn — can be looked upon as an approximation to $\mathbf{w}_{\mathrm{bp}}$ in the noise-free case [4]. Again, $\mathbf{w}_{\mathrm{bp}}$ is estimated by replacing the expectation by the mean over samples $\mathbf{w}_j$. Note that there exists no SVM equivalence $\mathbf{w}_{\mathrm{SVM}}$ to the Bayes point $\mathbf{w}_{\mathrm{bp}}$ in the case of label noise — a fact to be elaborated on in the experimental part in Section 5.

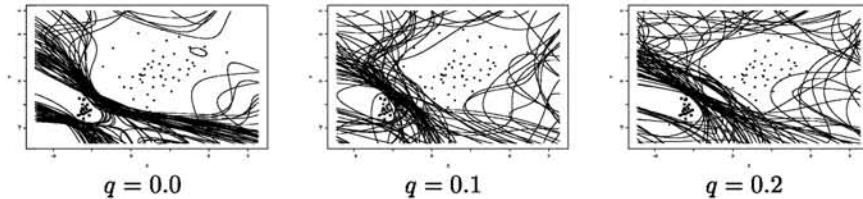

$$q = 0.0 \qquad\qquad q = 0.1 \qquad\qquad q = 0.2$$

Figure 3: A set of 50 samples $\mathbf{w}_j$ of the posterior $\mathbf{P}_{\mathbf{W}|\mathbf{Z}^m=\boldsymbol{z}}$ for various noise levels $q$. Shown are the resulting decision boundaries in data space $\mathcal{X}$.

**Active Learning**  The Bayesian posterior can also be employed to determine the usefulness of candidate training points — a task that can be considered as a dual counterpart to Bayesian Transduction. This is particularly useful when the label $y$ of a training point $x$ is more expensive to obtain than the training point $x$ itself. It was shown in the context of "Query by Committee" [2] that the binary entropy

$$S\left(x, z\right) = p^+ \log_2 p^+ + p^- \log_2 p^-$$

with $p^{\pm} = \mathbf{P}_{\mathbf{W}|\mathbf{Z}^m=\boldsymbol{z}}\left(\pm \langle \mathbf{W}, \mathbf{x} \rangle_{\mathcal{K}} > 0\right)$ is an indicator of the information content of a data point $x$ with regard to the learning task. Samples $\mathbf{w}_j$ from the Bayesian posterior $\mathbf{P}_{\mathbf{W}|\mathbf{Z}^m=\boldsymbol{z}}$ make it possible to estimate $S$ for a given candidate training points $x$ and the current training set $z$ to decide on the basis of $S$ if it is worthwhile to query the corresponding label $y$.

**Evidence Estimation for Model Selection**  Bayesian model selection is often based on a quantity called the *evidence* [5] of the model (given by the denominator of (2))

$$\mathbf{E}_{\mathbf{W}}\left[\mathbf{P}_{\mathbf{Y}^m|\mathbf{X}^m=\boldsymbol{x}, \mathbf{W}=\mathbf{w}}\left(y\right)\right] .$$

In the PAC-Bayesian framework this quantity has been demonstrated to be responsible for the generalisation performance of a model [6]. It turns out that in the zero-noise case the margin (the quantity maximised by the SVM) is a measure of the *evidence* of the model used [4]. In the case of label noise the KGS serves to estimate this quantity.

## 5  Experiments

In a first experiment we used a surrogate dataset of $m = 76$ data points $x$ in $\mathcal{X} = \mathbb{R}^2$ and the kernel $k\left(x, x'\right) = \exp(-\frac{1}{2} \|x - x'\|_{\mathcal{X}}^2)$. Using the KGS we sampled 50 different classifiers with weight vectors $\mathbf{w}_j$ for various noise levels $q$ and plotted the resulting decision boundaries $\left\{x \in \mathbb{R}^2 \ \middle|\ \langle \mathbf{w}_j, \mathbf{x} \rangle_{\mathcal{K}} = 0\right\}$ in Figure 3 (circles and crosses depict different classes). As can be seen form these plots, increasing the noise level $q$ leads to more diverse classifiers on the training set $z$.

In a second experiment we investigated the generalisation performance of the Bayes point machine (see (5)) in the case of label noise. In $\mathbb{R}^3$ we generated 100 random training and test sets of size $m_{\text{train}} = 100$ and $m_{\text{test}} = 1000$, respectively. For each normalised point $x \in \mathbb{R}^3$ the longitude and latitude were sampled from a Beta$(5, 5)$ and Beta$(0.1, 0.1)$ distribution, respectively. The classes $y$ were obtained by randomly flipping the classes assigned by the classifier $\mathbf{w}^*$ at $\left(\frac{\pi}{2}, \pi\right)$ (see also Figure 1) with a true label flip rate of $q^* = 5\%$. In Figure 4 we plotted the estimated generalisation error for a BPM (trained using 100 samples $\mathbf{w}_j$ from the KGS) and

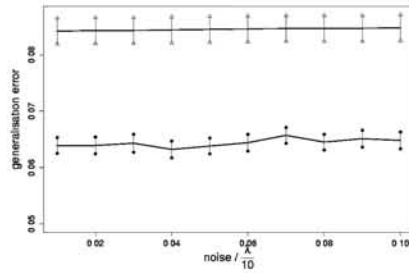 Generalisation errors of BPMs (circled error-bars) and soft-margin SVMs (triangled error-bars) vs. assumed noise level $q$ and margin slack penalisation $\lambda$, respectively. The dataset consisted of $m = 100$ observations with a label noise of 5% (dotted line) and we used $k(x,x') = \langle x, x' \rangle_\mathcal{X} + \lambda \cdot \mathbf{I}_{x=x'}$. Note that the abscissa is jointly used for $q$ and $\lambda$.

Figure 4: Comparison of BPMs and SVMs on data contaminated by label noise.

quadratic soft-margin SVM at different label noise levels $q$ and margin slack penalisation $\lambda$, respectively. Clearly, the BPM with the correct noise model outperformed the SVM irrespective of the chosen level of regularisation. Interestingly, the BPM appears to be quite "robust" w.r.t. the choice of the label noise parameter $q$.

## 6 Conclusion and Future Research

The kernel Gibbs sampler provides an analytical tool for the exploration of various Bayesian aspects of learning in kernel spaces. It provides a well-founded way for dealing with label noise but suffers from its computational complexity which — so far — makes it inapplicable for large scale applications. Therefore it will be an interesting topic for future research to invent new sampling schemes that may be able to trade accuracy for speed and would thus be applicable to large data sets.

**Acknowledgements**  This work was partially done while RH and TG were visiting Robert C. Williamson at the ANU Canberra. Thanks, Bob, for your great hospitality!

## Footnotes

[1]For the sake of convenience, we sometimes abbreviate $\phi(x)$ by $\mathbf{x}$. This, however, should not be confused with $n$–tuple $\boldsymbol{x}$ denoting the training objects.

## References

[1] C. Cortes and V. Vapnik. Support Vector Networks. *Machine Learning*, 20:273–297, 1995.

[2] Y. Freund, H. S. Seung, E. Shamir, and N. Tishby. Selective sampling using the query by committee algorithm. *Machine Learning*, 28:133–168, 1997.

[3] T. Graepel, R. Herbrich, and K. Obermayer. Bayesian Transduction. In *Advances in Neural Information System Processing 12*, pages 456–462, 2000.

[4] R. Herbrich, T. Graepel, and C. Campbell. Bayesian learning in reproducing kernel Hilbert spaces. Technical report, Technical University of Berlin, 1999. TR 99-11.

[5] D. MacKay. The evidence framework applied to classification networks. *Neural Computation*, 4(5):720–736, 1992.

[6] D. A. McAllester. Some PAC Bayesian theorems. In *Proceedings of the Eleventh Annual Conference on Computational Learning Theory*, pages 230–234, Madison, Wisconsin, 1998.

[7] R. M. Neal. Probabilistic inference using Markov chain Monte Carlo methods. Technical report, Dept. of Computer Science, University of Toronto, 1993. CRG-TR-93-1.

[8] P. Sollich. Probabilistic methods for Support Vector Machines. In *Advances in Neural Information Processing Systems 12*, pages 349–355, San Mateo, CA, 2000. Morgan Kaufmann.

[9] G. Wahba. Support Vector Machines, Reproducing Kernel Hilbert Spaces and the randomized GACV. Technical report, Department of Statistics, University of Wisconsin, Madison, 1997. TR–NO–984.